# Learning Lateral Interactions for Feature Binding and Sensory Segmentation

**Heiko Wersing**
HONDA R&D Europe GmbH
Carl-Legien-Str.30, 63073 Offenbach/Main, Germany
*heiko.wersing@hre-ftr.f.rd.honda.co.jp*

## Abstract

We present a new approach to the supervised learning of lateral interactions for the competitive layer model (CLM) dynamic feature binding architecture. The method is based on consistency conditions, which were recently shown to characterize the attractor states of this linear threshold recurrent network. For a given set of training examples the learning problem is formulated as a convex quadratic optimization problem in the lateral interaction weights. An efficient dimension reduction of the learning problem can be achieved by using a linear superposition of basis interactions. We show the successful application of the method to a medical image segmentation problem of fluorescence microscope cell images.

## 1 Introduction

Feature binding has been proposed to provide elegant solution strategies to the segmentation problem in perception [11, 12, 14]. A lot of feature binding models have thus tried to reproduce groping mechanisms like the Gestalt laws of visual perception, e.g. connectedness and good continuation, using temporal synchronization [12] or spatial coactivation [9, 14] for binding. Quite generally in these models, grouping is based on lateral interactions between feature-representing neurons, which characterize the degree of compatibility between features. Currently in most of the approaches this lateral interaction scheme is chosen heuristically, since the experimental data on the corresponding connection patterns in the visual cortex is insufficient. Nevertheless, in more complex feature spaces this heuristic approach becomes infeasible, raising the question for more systematic learning methods for lateral interactions.

Mozer et al. [4] suggested supervised learning for a dynamic feature binding model of complex-valued directional units, where the connections to hidden units guiding the grouping dynamics were adapted by recurrent backpropagation learning. The application was limited to synthetic rectangle patterns. Hofmann et al. [2] considered unsupervised texture segmentation by a pairwise clustering approach on feature vectors derived from Gabor filter banks at different frequencies and orientations. In their model the pairwise feature compatibilities are determined by a divergence measure of the local feature distributions which was shown to achieve good segmentation results for a range of image types. The problem of segmentation can also be phrased as a labeling problem, where relaxation labeling algorithms have been used as a popular tool in a wide range of computer vision applications.

Pelillo & Refice [7] suggested a supervised learning method for the compatibility coefficients of relaxation labeling algorithms, based on minimizing the distance between a target labeling vector and the output after iterating a fixed number of relaxation steps. The main problem are multiple local minima arising in this highly nonlinear optimization problem.

Recent results have shown that linear threshold (LT) networks provide interesting architectures for combining properties of digital selection and analogue context-sensitive amplification [1, 13] with efficient hardware implementation options [1]. Xie et al. [16] demonstrated how these properties can be used to learn winner-take-all competition between groups of neurons in an LT network with lateral inhibition. The CLM binding model is implemented by a large-scale topographically organized LT network, and it was shown that this leads to consistency conditions characterizing its binding states [14]. In this contribution we show how these conditions can be used to formulate a learning approach for the CLM as a quadratic optimization problem. In Section 2 we briefly introduce the competitive layer binding model. Our learning approach is elaborated in Section 3. In Section 4 we show application results of the approach to a cell segmentation problem and give a discussion in the final Section 5.

## 2 The CLM architecture

The CLM [9, 14] consists of a set of $L$ layers of feature-selective neurons (see Fig. 1). The activity of a neuron at position $r$ in layer $\alpha$ is denoted by $x_{r\alpha}$, and a *column* $r$ denotes the set of the neuron activities $x_{r\alpha}, \alpha = 1, \ldots, L$, sharing a common position $r$. With each column a particular "feature" is associated, which is described by a set of parameters like e.g. local edge elements characterized by position and orientation $(x_r, y_r, \theta_r)$. A binding between two features, represented by columns $r$ and $r'$, is expressed by simultaneous activities $x_{r\hat{\alpha}} > 0$ and $x_{r'\hat{\alpha}} > 0$ that share a common layer $\hat{\alpha}$. All neurons in a column $r$ are equally driven by an external input $h_r$, which represents the significance of the detection of feature $r$ by a preprocessing step. The afferent input $h_r$ is fed to the activities $x_{r\alpha}$ with a connection weight $J > 0$. Within each layer $\alpha$ the activities are coupled via lateral connections $f_{rr'}^{\alpha}$ which characterize the degree of compatibility between features $r$ and $r'$ and which is a symmetric function of the feature parameters, thus $f_{rr'}^{\alpha} = f_{r'r}^{\alpha}$. The purpose of the layered arrangement in the CLM is to enforce an assignment of the input features to the layers by the dynamics, using the contextual information stored in the lateral interactions. The unique assignment to a single layer is realized by a columnar Winner-Take-All (WTA) circuit, which uses mutual symmetric inhibitory interactions with absolute strength $J > 0$ between neural activities $x_{r\alpha}$ and $x_{r\beta}$ that share a common column $r$. Due to the WTA coupling, for a stable equilibrium state of the CLM only a neuron from one layer can be active within each column [14]. The number of layers does not predetermine the number of active groups, since for sufficiently many layers only those are active that carry a salient group. The combination of afferent inputs and lateral and vertical interactions is combined into the standard linear threshold additive activity dynamics

$$\dot{x}_{r\alpha} = -x_{r\alpha} + \sigma\Big(J(h_r - \sum_\beta x_{r\beta}) + \sum_{r'} f_{rr'}^{\alpha} x_{r'\alpha}\Big), \tag{1}$$

where $\sigma(x) = \max(0, x)$. For $J$ large compared to the lateral weights $f_{rr'}^{\alpha}$, the single active neuron in a column reproduces its afferent input, $x_{r\alpha} \approx h_r$. As was shown [14], the stable states of (1) satisfy the consistency conditions

$$\sum_{r'} f_{rr'}^{\beta} x_{r'\beta} < \sum_{r'} f_{rr'}^{\hat{\alpha}} x_{r'\hat{\alpha}} \quad \text{for all} \quad r, \beta \neq \hat{\alpha}(r), \tag{2}$$

which express the assignment of a feature $r$ to the layer $\hat{\alpha}(r)$ with highest lateral support.

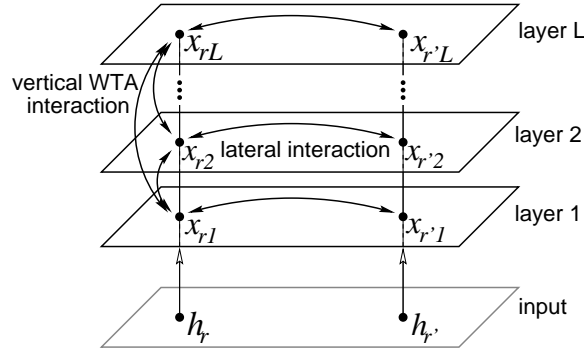

Figure 1: The competitive layer model architecture (see text for description).

## 3 Learning of CLM Lateral Interactions

**Formulation of the Learning Problem.** The overall task of the learning algorithm is to adapt the lateral interactions, given by the interaction coefficients $\mathsf{f}_{rr'}^{\alpha}$, such that the CLM architecture performs appropriate segmentation on the labeled training data and also generalizes to new test data. We assume that the training data consists of a set of $M$ labeled training patterns $\mathcal{P}^i$, $i = 1, \ldots, M$, where each pattern $\mathcal{P}^i$ consists of a subset $\mathcal{R}^i = \{r_1^i, \ldots, r_{N^i}^i\}$ of $N^i$ different features with their corresponding labels $\hat{\alpha}^i(r_j)$. For each labeled training pattern a target labeling vector $\mathbf{y}^i$ is constructed by choosing

$$y_{r\hat{\alpha}^i(r)}^i = 1, \quad y_{r\beta}^i = 0 \qquad \text{for all} \quad r \in \mathcal{R}^i, \; \beta \neq \hat{\alpha}^i(r) \tag{3}$$

for the labeled columns, assuming $h_r = 1$. Columns for features which are not contained in the training pattern are filled with zeroes according to $y_{s\alpha}^i = 0$ for all $\alpha, s \notin \mathcal{R}^i$. In the following indices $s, s'$ run over all possible $N$ features, e.g. all edges of different orientations at different image positions, while $r, r'$ run over the subset of features realized in a particular pattern, e.g. only one oriented edge at each image position. The assignment vectors $\mathbf{y}^i$, $i = 1, \ldots, M$ form the basis of the learning approach since they represent the target activity distribution, which we want to obtain after iterating the CLM with appropriately adjusted lateral interactions. In the following the abbreviation $\hat{\alpha}$ for $\hat{\alpha}^i(r)$ is used to keep the notation readable.

The goal of the learning process is to make the training patterns consistent, which is in accordance with (2) expressed by the inequalities

$$\sum_{r'} \mathsf{f}_{rr'}^{\beta} y_{r'\beta}^i < \sum_{r'} \mathsf{f}_{rr'}^{\hat{\alpha}} y_{r'\hat{\alpha}}^i \quad \text{for all} \quad i, r \in \mathcal{R}^i, \beta \neq \hat{\alpha}. \tag{4}$$

These $(L-1)\sum_i N^i$ inequalities define the learning problem that we want to solve in the following. Let us develop a more compact notation. We can rewrite (4) as

$$\sum_{\gamma s s'} Y_{\gamma s s'}^{ri\beta} \mathsf{f}_{ss'}^{\gamma} < 0 \quad \text{for all} \quad i, r \in \mathcal{R}^i, \beta \neq \hat{\alpha}, \tag{5}$$

where $Y_{\gamma s s'}^{ir\beta} = \delta_{rs}\left(\delta_{\beta\gamma}y_{s'\beta}^i - \delta_{\hat{\alpha}\gamma}y_{s'\hat{\alpha}}^i\right)$. The form of the inequalities can be simplified by introducing multiindices $\mathbf{k}$ and $\boldsymbol{\nu}$ which correspond to $\boldsymbol{\nu} \leftrightarrow (\gamma, s, s')$, $\mathbf{k} \leftrightarrow (i, r, \beta)$ and $\mathsf{f}_{\boldsymbol{\nu}} \leftrightarrow \mathsf{f}_{ss'}^{\gamma}$, $Y_{\boldsymbol{\nu}}^{\mathbf{k}} \leftrightarrow Y_{\gamma s s'}^{ir\beta}$. The index $\mathbf{k}$ runs over all $(L-1)\sum_i N^i$ consistency relations defined for the labeled columns of the assignment vectors. The vectors $\mathbf{Y}^{\mathbf{k}}$ with components $Y_{\boldsymbol{\nu}}^{\mathbf{k}}$ are called *consistency vectors* and represent the consistency constraints for the lateral interaction. The index $\boldsymbol{\nu}$ runs over all entries in the lateral interaction matrix. The vector

$\mathbf{f} = (\mathsf{f}^1_{11}, \ldots, \mathsf{f}^1_{NN}, \ldots, \mathsf{f}^L_{11}, \ldots, \mathsf{f}^L_{NN})$ with $LN^2$ components contains the corresponding matrix entries. The inequalities (4) can then be written in the form

$$\sum_{\boldsymbol{\nu}} Y^{\mathbf{k}}_{\boldsymbol{\nu}} \mathsf{f}_{\boldsymbol{\nu}} < 0 \quad \text{for all} \quad \mathbf{k}. \tag{6}$$

This illustrates the nature of the learning problem. The problem is to find a weight vector $\mathbf{f}$ which leads to a lateral interaction matrix, such that the consistency vectors lie in the opposite half space of the weight state space. Since the conditions (6) determine the attractivity of the training patterns, it is customary to introduce a positive margin $\kappa > 0$ to achieve greater robustness. This gives the target inequalities

$$\sum_{\boldsymbol{\nu}} Y^{\mathbf{k}}_{\boldsymbol{\nu}} \mathsf{f}_{\boldsymbol{\nu}} + \kappa < 0 \quad \text{for all} \quad \mathbf{k}, \tag{7}$$

which we want to solve in $\mathbf{f}$ for given training data. If the system of inequalities admits a solution for $\mathbf{f}$ it is called *compatible*. If there is no $\mathbf{f}$ satisfying all constraints, the system is called *incompatible*.

**Superposition of Basis Interactions.** If the number of features $N$ is large, the number of parameters in the complete interaction matrix $\mathsf{f}^{\gamma}_{ss'}$ may be too large to be robustly estimated from a limited number of training examples. To achieve generalization from the training data, it is necessary to reduce the number of parameters which have to be adapted during learning. This is also useful to incorporate a priori knowledge into the interaction. An example is to choose basis functions which incorporate invariances such as translation and rotation invariance, or which satisfy the constraint that the interaction is equal in all layers. A simple but powerful approach is to choose a set of $K$ fixed basis interactions $g^j$ with compatibilities $g^{j\gamma}_{ss'}$, $\qquad j = 1, \ldots, K$, with an interaction $\mathsf{f}^{\gamma}_{ss'}$ obtained by linear superposition

$$\mathsf{f}^{\gamma}_{ss'} = \sum_{j} c_j g^{j\gamma}_{ss'} = \sum_{j} c_j g^{j}_{\boldsymbol{\nu}} \tag{8}$$

with weight coefficients $c_j, j = 1, \ldots, K$. Now the learning problem of solving the inequalities (7) can be recast in the new free parameters $c_j$. After inserting (8) into (7) we obtain the transformed problem

$$\sum_{\boldsymbol{\nu}} Y^{\mathbf{k}}_{\boldsymbol{\nu}} \sum_{j} c_j g^{j}_{\boldsymbol{\nu}} + \kappa = \sum_{j} c_j Z^{\mathbf{k}}_{j} + \kappa < 0 \quad \text{for all} \quad \mathbf{k}, \tag{9}$$

where $Z^{\mathbf{k}}_{j} = \sum_{\boldsymbol{\nu}} Y^{\mathbf{k}}_{\boldsymbol{\nu}} g^{j}_{\boldsymbol{\nu}}$ is the component of the consistency vector $\mathbf{Y}^{\mathbf{k}}$ in the basis interaction $g^j$. The basis interactions can thus be used to reduce the dimensionality of the learning problem. To avoid any redundancy, the basis interactions should be linearly independent. Although the functions are here denoted "basis" functions, they need neither be orthogonal nor span the whole space of interactions $\mathbf{f} \in R^{LN^2}$.

**Quadratic Consistency Optimization.** The generic case in any real world application is that the majority of training vectors contains relevant information, while single spurious vectors may be present due to noise or other disturbing factors. Consequently, in most applications the equations (7) or (9) will be incompatible and can only be satisfied approximately. This will be especially the case, if a low-dimensional embedding is used for the basis function templates as described above. We therefore suggest to adapt the interactions by minimizing the following convex cost function

$$E^{\text{QCO}} = \sum_{\mathbf{k}} \left( \sum_{\boldsymbol{\nu}} Y^{\mathbf{k}}_{\boldsymbol{\nu}} \mathsf{f}_{\boldsymbol{\nu}} + \kappa \right)^2. \tag{10}$$

A similar minimization approach was suggested for the imprinting of attractors for the Brain-State-in-a-Box (BSB) model [8], and a recent study has shown that the approach is competitive with other methods for designing BSB associative memories [6].

For a fixed positive margin $\kappa > 0$, the cost function (10) is minimized by making the inner products of the weight vector and the consistency vectors negative. The global minimum with $E^{\mathrm{QCO}} = 0$ is attained if the inner products are all equal to $-\kappa$, which can be interpreted such that all consistency inequalities are fulfilled in an equal manner. Although this additional regularizing constraint is hard to justify on theoretical grounds, the later application shows that it works quite well for the application examples considered.

If we insert the expansion of $\mathbf{f}$ in the basis of function templates we obtain according to (8)

$$E^{\mathrm{QCO}} = \sum_{\mathbf{k}} \left( \sum_j c_j Z_j^{\mathbf{k}} + \kappa \right)^2, \tag{11}$$

which results in a $K$-dimensional convex quadratic minimization problem in the $c_j$ parameters. The coefficients $Z_j^{\mathbf{k}}$, which give the components of the training patterns in the basis interactions, are given by $Z_j^{\mathbf{k}} = \sum_{\boldsymbol{\nu}} Y_{\boldsymbol{\nu}}^{\mathbf{k}} g_{\boldsymbol{\nu}}^j = \sum_{r'} y_{r'\beta}^i g_{rr'}^{j\beta} - \sum_{r'} y_{r'\hat{\alpha}}^i g_{rr'}^{j\hat{\alpha}}$. The quadratic optimization problem is then given by minimizing

$$E^{\mathrm{QCO}} = \sum_{jl} A_{jl} c_j c_l + \sum_j B_j c_j + \kappa^2, \tag{12}$$

where $A_{jl} = \sum_{\mathbf{k}} Z_j^{\mathbf{k}} Z_l^{\mathbf{k}}$ and $B_j = 2\kappa \sum_{\mathbf{k}} Z_j^{\mathbf{k}}$. If the coefficients $c_j$ are unconstrained, then the minimum of (12) can be obtained by solving the linear system of equations $\partial E / \partial c_j = 2 \sum_l A_{jl} c_l + B_j = 0$ for all $j$.

## 4 Application to Cell Segmentation

The automatic detection and segmentation of individual cells in fluorescence micrographs is a key technology for high-throughput analysis of immune cell surface proteins [5]. The strong shape variability of cells in tissue, however, poses a strong challenge to any automatic recognition approach. Figure 2a shows corresponding fluorescence microscopy images from a tissue section containing lymphocyte cells (courtesy W. Schubert). In the bottom row corresponding image patches are displayed, where individual cell regions were manually labeled to obtain training data for the learning process.

For each of the image patches, a training vector consists of a list of labeled edge features parameterized by $(\mathbf{p}_r, \mathbf{n}_r)$, where $\mathbf{p}_r$ is the position in the image and $\mathbf{n}_r$ is a unit local edge orientation vector computed from the intensity gradient. For a $40 \times 40$ pixel image this amounts to a set of $40^2$ labeled edge features. Since the figure-ground separating mechanism as implemented by the CLM [14] is also used for this cell segmentation application, features which are not labeled as part of a cell obtain the corresponding background label, given by $\alpha = 1$. Each training pattern contains one additional free layer, to enable the learning algorithm to generalize over the number of layers.

The lateral interaction to be adapted is decomposed into the following weighted basis components: i) A constant negative interaction between all features, which facilitates group separation, ii) a self-coupling interaction in the background layer which determines the attractivity of the background for figure-ground segmentation, and iii) an angular interaction with limited range, which is in itself decomposed into templates, capturing the interaction for a particular combination of the relative angles between two edges. This angular decomposition is done using a discretization of the space of orientations, turning the unit-vector representation into an angular orientation variable $\theta \in [0, 2\pi[$. To achieve rotation invariance of the interaction, it is only dependent on the edge orientations relative

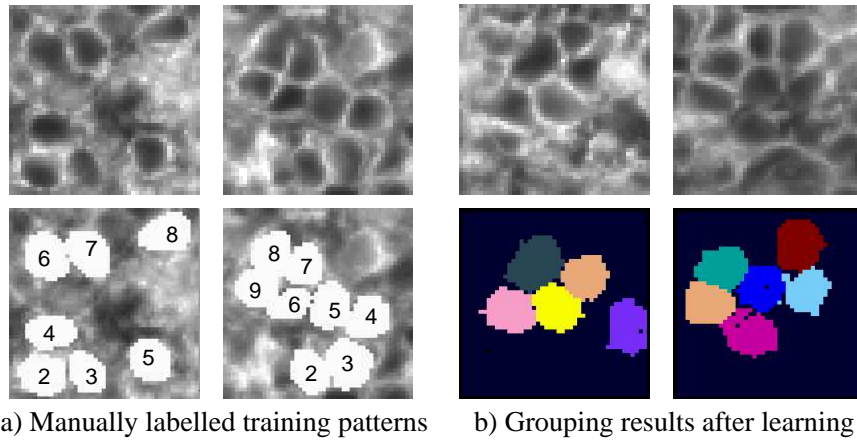

| a) Manually labelled training patterns | b) Grouping results after learning |

Figure 2: a) Original images and manually labeled training patterns from a fluorescence micrograph. b) Test patterns and resulting CLM segmentation with learned lateral interaction. Grayscale represents different layer activations, where a total of 20 layers plus one background layer (black) was used.

to their mutual position difference vector $\mathbf{d} = \mathbf{p}_2 - \mathbf{p}_1$. The angles $\theta_1$ and $\theta_2$ are discretized by partitioning the interval $[0, 2\pi[$ into 8 subintervals. For each combination of the two discretized edge orientations there is an interaction template generated, which is only responding in this combined orientation interval. Thus the angular templates do not overlap in the combined $(\theta_1, \theta_2)$ space, i.e. if $g^{j\gamma}((\mathbf{p}_1, \theta_1), (\mathbf{p}_2, \theta_2)) = 1$ for a particular $j$, then $g^{l\gamma}((\mathbf{p}_1, \theta_1), (\mathbf{p}_2.\theta_2)) = 0$ for all $l \neq j$. Since the interaction must be symmetric under feature exchange, this does not result in $8^2 = 64$ different combinations, but only 36 independent templates. Apart form the discretization, the interaction represents the most arbitrary angular-dependent interaction within the local neighborhood, which is symmetric under feature exchange. We use two sets of angular templates for $|\mathbf{d}| < R/2$ and $R/2 < |\mathbf{d}| < R$ respectively, where $R$ is the maximal local interaction range. With the abovementioned two components, the resulting optimization problem is 36+36+2=74-dimensional. Figure 3 compares the optimized interaction field to earlier heuristic lateral interactions for contour grouping. See [15] for a more detailed discussion.

The performance of the learning approach was investigated by choosing a small number of the manually labeled patterns as training patterns. For all the training examples we used, the resulting inequalities (9) were in fact incompatible, rendering a direct solution of (9) infeasible. After training was completed by minimizing (12), a new image patch was selected as a test pattern and the CLM grouping was performed with the lateral interaction learned before, using the dynamical model as described in [14]. The quadratic consistency optimization was performed as described in the previous section, exploring the free margin parameter $\kappa$. For a set of two training patterns as shown in Fig. (2)a with a total of 1600 features each, a learning sweep takes about 4 minutes on a standard desktop computer.

Typical segmentation results obtained with the quadratic consistency optimization approach are shown in Figure 2b, where the margin was given by $\kappa = 10$. The grouping results were not very sensitive to $\kappa$ in a range of $5 < \kappa < 100$. The grouping results show a good segmentation performance where most of the salient cells are detected as single groups. There are some spurious groups where a dark image region forms an additional group and some smaller cells are rejected into the background layer. Apart from these minor errors, the optimization has achieved an adequate balancing of the different lateral interaction components for this segmentation task.

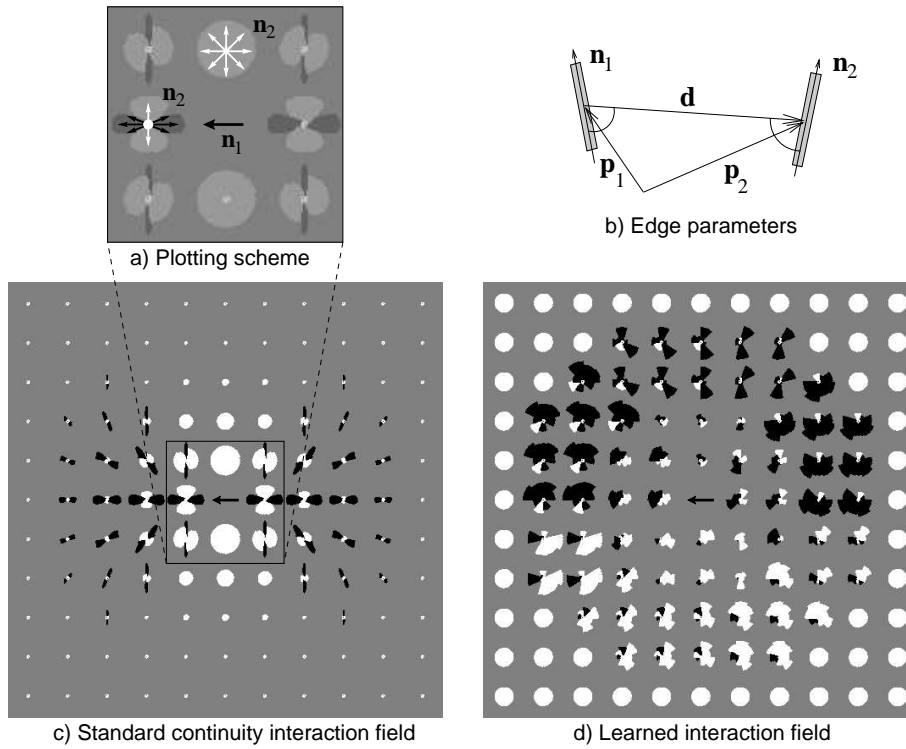

a) Plotting scheme

b) Edge parameters

c) Standard continuity interaction field

d) Learned interaction field

Figure 3: Comparison between heuristic continuity grouping interaction field and a learned lateral interaction field for cell segmentation. The interaction depends on the difference vector $\mathbf{d}$ and two unit vectors $\mathbf{n}_1, \mathbf{n}_2$, shown in b), encoding directed orientation. a) explains the interaction visualizations c) and d) by showing a magnification of the plot c) of the interaction field of a single horizontal edge pointing to the left. The plots are generated by computing the interaction of the central directed edge with directed edges of all directions (like a cylindrical plot) at a spatial grid. Black edges share excitatory, white edges share inhibitory interaction with the central edge and length codes for interaction strength. The cocircular continuity field in c) depends on position and orientation but is not direction-selective. It supports pairs of edges which are cocircular, i.e. lie tangentially to a common circle and has been recently used for contour segmentation [3, 14]. The learned lateral interaction field is shown in d). It is direction-selective and supports pairs of edges which "turn right". The strong local support is balanced by similarly strong long-range inhibition.

## 5 Discussion

The presented results show that appropriate lateral interactions can be obtained for the CLM binding architecture from the quadratic consistency optimization approach. The only a priori conditions which were used for the template design were the properties of locality, symmetry, and translation as well as rotation invariance. This supervised learning approach has clear advantages over the manual tuning of complex feature interactions in complex feature spaces with many parameters. We consider this as an important step towards practical applicability of the feature binding concept.

The presented quadratic consistency optimization method is based on choosing equal margins for all consistency inequalities. There exist other approaches to large margin classifica-

tion, like support vector machines [10], where more sophisticated methods were suggested for appropriate margin determination. The application of similar methods to the supervised learning of CLM interactions provides an interesting field for future work.

**Acknowledgments:** This work was supported by DFG grant GK-231 and carried out at the Faculty of Technology, University of Bielefeld. The author thanks Helge Ritter and Tim Nattkemper for discussions and Walter Schubert for providing the cell image data.

## References

[1] R. Hahnloser, R. Sarpeshkar, M. A. Mahowald, R. J. Douglas, and H. S. Seung. Digital selection and analogue amplification coexist in a cortex-inspired silicon circuit. *Nature*, 405:947–951, 2000.

[2] T. Hofmann, J. Puzicha, and J. Buhmann. Unsupervised texture segmentation in a deterministic annealing framework. *IEEE Trans. Pattern Analysis and Machine Intelligence*, 20(8):803–818, 1998.

[3] Z. Li. A neural model of contour integration in the primary visual cortex. *Neural Computation*, 10:903–940, 1998.

[4] M. Mozer, R. S. Zemel, M. Behrmann, and C. K. I. Williams. Learning to segment images using dynamic feature binding. *Neural Computation*, 4(5):650–665, 1992.

[5] T. W. Nattkemper, H. Ritter, and W. Schubert. A neural classificator enabling high-throughput topological analysis of lymphocytes in tissue sections. *IEEE Trans. Inf. Techn. in Biomed.*, 5(2):138–149, 2001.

[6] J. Park, H. Cho, and D. Park. On the design of BSB associative memories using semidefinite programming. *Neural Computation*, 11:1985–1994, 1999.

[7] M. Pelillo and M Refice. Learning compatibility coefficients for relaxation labeling processes. *IEEE Trans. Pattern Analysis and Machine Intelligence*, 16(9):933–945, 1994.

[8] Renzo Perfetti. A synthesis procedure for Brain-State-in-a-Box neural networks. *IEEE Transactions on Neural Networks*, 6(5):1071–1080, September 1995.

[9] H. Ritter. A spatial approach to feature linking. In *Proc. International Neural Network Conference Paris Vol.2*, pages 898–901, 1990.

[10] V. Vapnik. *The nature of statistical learning theory*. Springer, New York, 1995.

[11] C. von der Malsburg. The what and why of binding: The modeler's perspective. *Neuron*, 24:95–104, 1999.

[12] D. Wang and D. Terman. Image segmentation based on oscillatory correlation. *Neural Computation*, 9(4):805–836, 1997.

[13] H. Wersing, W.-J. Beyn, and H. Ritter. Dynamical stability conditions for recurrent neural networks with unsaturating piecewise linear transfer functions. *Neural Computation*, 13(8):1811–1825, 2001.

[14] H. Wersing, J. J. Steil, and H. Ritter. A competitive layer model for feature binding and sensory segmentation. *Neural Computation*, 13(2):357–387, 2001.

[15] Heiko Wersing. *Spatial Feature Binding and Learning in Competitive Neural Layer Architectures*. PhD thesis, University of Bielefeld, 2000. Published by Cuvillier, Goettingen.

[16] X. Xie, R. Hahnloser, and H.S. Seung. Learning winner-take-all competition between groups of neurons in lateral inhibition networks. In *Advances in Neural Information Processing Systems*, volume 13. The MIT Press, 2001.